# Dynamically-Adaptive Winner-Take-All Networks

**Trent E. Lange**
Artificial Intelligence Laboratory
Computer Science Department
University of California, Los Angeles, CA 90024

## Abstract

Winner-Take-All (WTA) networks, in which inhibitory interconnections are used to determine the most highly-activated of a pool of units, are an important part of many neural network models. Unfortunately, convergence of normal WTA networks is extremely sensitive to the magnitudes of their weights, which must be hand-tuned and which generally only provide the right amount of inhibition across a relatively small range of initial conditions. This paper presents *Dynamically-Adaptive Winner-Take-All (DAWTA)* networks, which use a regulatory unit to provide the competitive inhibition to the units in the network. The DAWTA regulatory unit dynamically adjusts its level of activation during competition to provide the right amount of inhibition to differentiate between competitors and drive a single winner. This dynamic adaptation allows DAWTA networks to perform the winner-take-all function for nearly any network size or initial condition, using $O(N)$ connections. In addition, the DAWTA regulatory unit can be biased to find the level of inhibition necessary to settle upon the $K$ most highly-activated units, and therefore serve as a K-Winners-Take-All network.

## 1. INTRODUCTION

Winner-Take-All networks are fixed group of units which compete by mutual inhibition until the unit with the highest initial activation or input level suppresses the activation of all the others. Winner-take-all selection of the most highly-activated unit is an important part of many neural network models (e.g. McClelland and Rumelhart, 1981; Feldman and Ballard, 1982; Kohonen, 1984; Touretzky, 1989; Lange and Dyer, 1989a,b).

Unfortunately, successful convergence in winner-take-all networks is extremely sensitive to the magnitudes of the inhibitory weights between units and other network parameters. For example, a weight value for the mutually-inhibitory connections allowing the most highly-activated unit to suppress the other units in one initial condition (e.g. Figure 1a) may not provide enough inhibition to select a single winner if the initial input activation levels are closer together and/or higher (e.g. Figure 1b). On the other hand, if the compe-

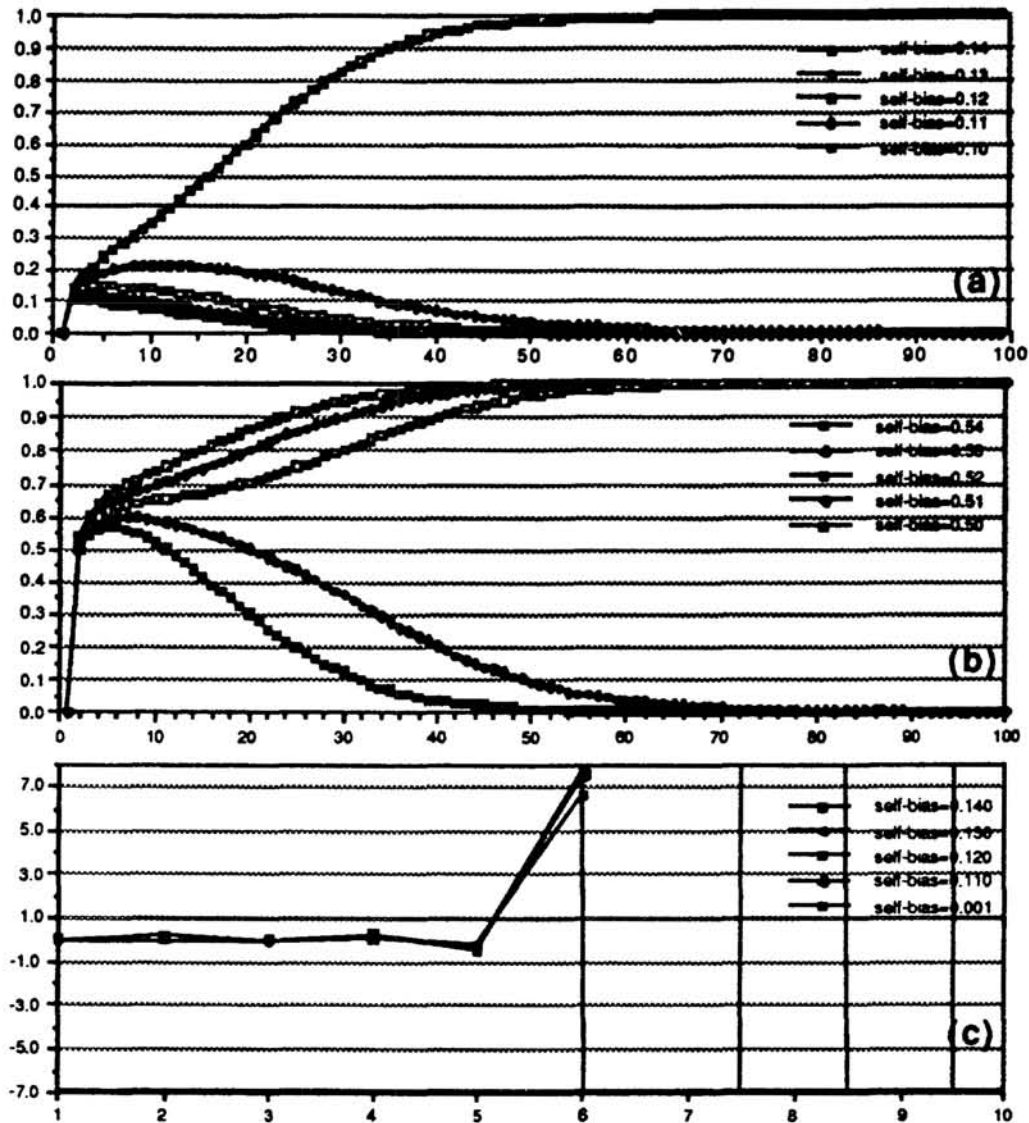

Figure 1. Several plots of activation versus time for different initial conditions in a winner-take-all network in which there is a bidirectional inhibitory connection of weight -0.2 between every pair of units. Unit activation function is that from the interactive activation model of McClelland and Rumelhart (1981). (a) Network in which five units are given an input self bias ranging from 0.10 to 0.14. (b) Network in which five units are given an input self bias ranging from 0.50 to 0.54. Note that the network ended up with three winners because the inhibitory connections of weight -0.2 did not provide enough inhibition to suppress the second and third most-active nodes. (c) Network in which 100 units are given an input self bias ranging from 0.01 to 0.14. The combined activation of all 100 nodes through the inhibitory weight of -0.2 provides far too much inhibition, causing the network to overreact and oscillate wildly

tition involves a larger number of active units, then the same inhibitory weights may provide too much inhibition and either suppress the activations of all units or lead to oscillations (e.g. Figure 1c).

Because of these problems, it is generally necessary to hand-tune network parameters to allow for successful winner-take-all performance in a given neural network architecture having certain expected levels of incoming activations. For complex networks, this can require a detailed mathematical analysis of the model (cf. Touretzky & Hinton, 1988) or a heuristic, computer-assisted trial-and-error search process (cf. Reggia, 1989) to find the values of inhibitory weights, unit thresholds, and other network parameters necessary for clear-cut winner-take-all performance in a given model's input space. In some cases, however, no set of network constant network parameters can be found to handle the range of possible initial conditions a model may be faced with (Barnden, Kankanahalli, and Dharmavaratha, 1990), such as when the numbers of units actually competing in a given network may be two at one time and thousands at another (e.g. Barnden, 1990; Lange, in press).

This paper presents a new variant of winner-take-all networks, the *Dynamically-Adaptive Winner-Take-All (DAWTA)* network. DAWTA networks, using $O(N)$ connections, are able to robustly act as winner-take-all networks for nearly any network initial condition without any hand-tuning of network parameters. In essence, the DAWTA network dynamically "tunes" itself by adjusting the level of inhibition sent to each unit in the network depending upon feedback from the current conditions of the competition. In addition, a biasing activation can be added to the network to allow it to act as a $K$-Winners-Take-All network (cf. Majani, Erlanson, and Abu-Mostafa, 1989), in which the $K$ most highly-activated units end up active.

## 2. DYNAMICALLY-ADAPTIVE WTA NETWORKS

The basic idea behind the Dynamically-Adaptive Winner-Take-All mechanism can be described by looking at a version of a winner-take-all network that is functionally equivalent to a normal winner-take-all network but which uses only $O(N)$ connections. Several researchers have pointed out that the $(N^2-N)/2$ bidirectional inhibitory connections (each of weight $-w_I$) normally needed in a winner-take-all network can be replaced by an excitatory self-connection of weight $w_I$ for each unit and a single regulatory unit that sums up the activations of all $N$ units and inhibits them each by that $-w_I$ times that amount (Touretzky & Hinton, 1988; Majani *et al.*, 1989) (see Figure 2).

When viewed in this fashion, the mutually inhibitory connections of winner-take-all networks can be seen as a regulator (i.e. the regulatory unit) that is attempting to provide the right amount of inhibition to the network to allow the winner-to-be unit's activation to grow while suppressing the activations of all others. This is exactly what happens when $w_I$ has been chosen correctly for the activations of the network (as in Figure 1a). However, because the amount of this regulatory inhibition is fixed precisely by that inhibitory weight (i.e. always equal to that weight times the sum of the network activations), there is no way for it to increase when it is not enough (as in Figure 1b) or decrease when it is too much (as in Figure 1c).

### 2.1. THE DAWTA REGULATORY UNIT

From the point of view of the competing units' inputs, the Dynamically-Adaptive Winner-Take-All network is equivalent to the regulatory-unit simplification of a normal winner-take-all network. Each unit has an excitatory connection to itself and an inhibitory connection from a regulatory unit whose function is to suppress the activations

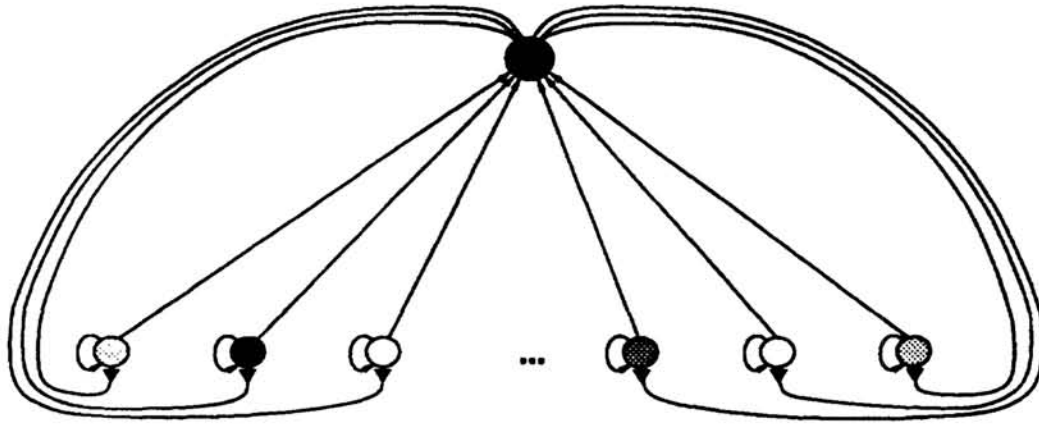

Figure 2. Simplification of a standard WTA network using $O(n)$ connections by introduction of a regulatory unit (top node) that sums up the activations of all network units. Each unit has an excitatory connection to itself and an inhibitory connection of weight $-w_I$ from the regulatory unit. Shading of units (darker = higher) represents their levels of activation at a hypothetical time in the middle of network cycling.

of all but the winning unit[1]. However, the regulatory unit itself, and how it calculates the inhibition it provides to the network, is different.

Whereas the connections to the regulatory unit in a normal winner-take-all network cause it to produce an inhibitory activation (i.e. the sum of the units' activations) that happens to work if its inhibitory weights were set correctly, the structure of connections to the regulatory unit in a dynamically-adaptive winner-take-all network cause it to continually adjust its level of activation until the right amount of inhibition is found, regardless of the network's initial conditions. As the network cycles and the winner-take-all is being performed, the DAWTA regulatory unit's activation inhibits the networks' units, which results in feedback to the regulatory unit that causes it to increase its activation if more inhibition is required to induce a single winner, or decrease its activation if less is required. Accordingly, the DAWTA regulatory unit's activation ($a_R(t)$) now includes its previous activation, and is the following:

$$a_R(t+1) = a_R(t) + \begin{cases} -\Theta & net_R(t+1) \leq -\Theta \\ net_R(t+1) & -\Theta < net_R(t+1) < \Theta \\ \Theta & net_R(t+1) \geq \Theta \end{cases}$$

where $net_R(t+1)$ is the total net input to the regulator at time $t+1$, and $\Theta$ is a small constant (typically 0.05) whose purpose is to stop the regulatory unit's activation from rising or falling too rapidly on any given cycle. Figure 3 shows the actual Dynamically-Adaptive Winner-Take-All network. As in Figure 2, the regulatory unit is the unit at the top and the competing units are the the circular units at the bottom that are inhibited by it and which have connections (of weight $w_s$) to themselves. However, there are now two

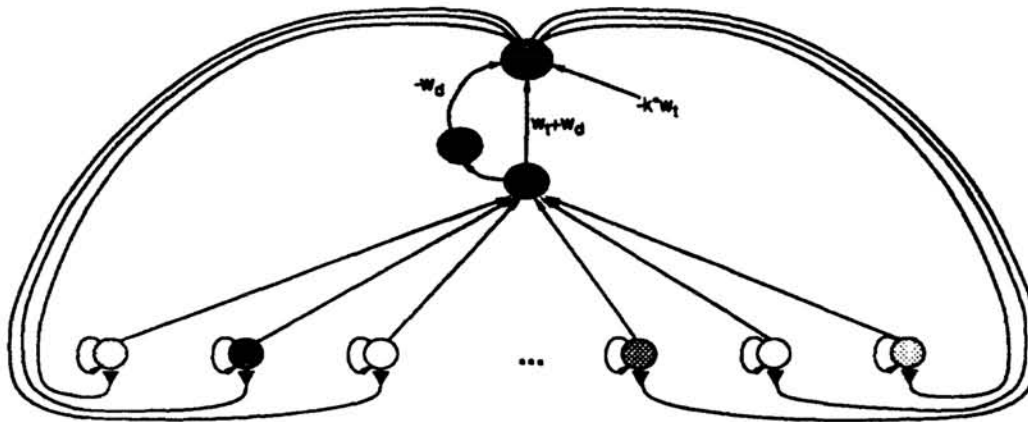

Figure 3. Dynamically-Adaptive Winner-Take-All Network at a hypothetical time in the middle of network cycling. The topmost unit is the DAWTA regulatory unit, whose outgoing connections to all of the competing units at the bottom all have weight -1. The input $-k^*w_d$ is a constant self biasing activation to the regulatory unit whose value determines how many winners it will try to drive. The two middle units are simple linear summation units each having inputs of unit weight that calculate the total activation of the competing units at time $t$ and time $t$-1, respectively.

intermediate units that calculate the net inputs that increase or decrease the regulatory unit's inhibitory activation depending on the state of the competition. These inputs cause the regulatory unit to receive a net input $net_R$ ($t$+1) of:

$$net_R(t+1) = (w_t + w_d)o_t(t-1) - w_d o_t(t-2) - kw_t$$

which simplifies to:

$$net_R(t+1) = w_t(o_t(t-1) - k) + w_d(o_t(t-1) - o_t(t-2))$$

where $o_t(t)$ is the total summed output of all of the competing units (calculated by the intermediate units shown), $w_t$ and $w_d$ are constant weights, and $k$ is the number of winners the network is attempting to seek (1 to perform a normal winner-take-all).

The effect of the above activation function and the connections shown in Figure 3 is to apply two different activation pressures on the regulatory unit, each of which combined over time drive the DAWTA regulatory unit's activation to find the right level of inhibition to suppress all but the winning unit. The most important pressure, and the key to the DAWTA regulatory unit's success, is that the regulatory unit's activation increases by a factor of $w_t$ if there is too much activation in the network, and decreases by a corresponding factor if there is not enough activation in the network. This is the result of the term $w_t(o_t(t-1) - k)$ in its net input function, which simplifies to $w_t(o_t(t-1) - 1)$ when $k$ equals 1. The "right amount" of total activation in the network is simply the total summed activation of the goal state, i.e. the winner-take-all network state in which there is one active unit (having activation 1) and in which all other competing units have

been driven down to an activation of 0, leaving the total network activation $o_t(t)$ equal to 1. The factor $w_t(o_t(t-1) - 1)$ of the regulatory input's net input will therefore tend to increase the regulatory unit's activation if there are too many units active in the network (e.g. if there are three units with activity 0.7, 0.5, and 0.3, since the total output $o_t(t)$ will be 1.5), to decrease its activation if there is not enough totally active units in the network (e.g. one unit with activation 0.2 and the rest with activation 0.0), and to leave its activation unchanged if the activation is the same as the final goal activation. Note that any temporary coincidences in which the total network activation sums to 1 but which is not the final winner-take-all state (e.g. when one unit has activation 0.6 and another has activation 0.4) will be broken by the competing units themselves, since the winning unit's activation will always rise more quickly than the loser's just by its own activation function (e.g. that of McClelland and Rumelhart, 1981).

The other pressure on the DAWTA regulatory unit, from the $w_d(o_t(t-1) - o_t(t-2))$ term of $net_R(t+1)$, is to tend to decrease the regulator's activation if the overall network activation is falling too rapidly, or to increase it if the overall network activation is rising too rapidly. This is essentially a dampening term to avoid oscillations in the network in the early stages of the winner-take-all, in which there may be many active units whose activations are falling rapidly (due inhibition from the regulatory unit), but in which the total network activation is still above the final goal activation. As can be seen, this second term of the regulatory unit's net input will also sum to 0 and therefore leave the regulatory unit's activation unchanged when the goal state of the network has been reached, since the total activation of the network in the winner-take-all state will remain constant.

All of the weights and connections of the DAWTA network are constant parameters that are the same for any size network or set of initial network conditions. Typically we have used $w_t = 0.025$ and $w_d = 0.5$. The actual values are not critical, as long as $w_d \gg w_s$, which assures that $w_d$ is high enough to dampen the rapid rise or fall in total network activation sometimes caused by the direct pressure of $w_t$. The value of the regulatory unit's self bias term $kw_t$ that sets the goal total network activation that the regulatory unit attempts to reach is simply determined simply by $k$, the number of winners desired (1 for a normal winner-take-all network), and $w_t$.

## 3. RESULTS

Dynamically-adaptive winner-take-all networks have been tested in the DESCARTES connectionist simulator (Lange, 1990) and used in our connectionist model of short-term sequential memory (Lange, in press). Figures 4a-c show the plots of activation versus time in networks given the same initial conditions as those of the normal winner-take-all network shown in Figures 1a-c. Note that in each case the regulatory unit's activation starts off at zero and increases until it reaches a level that provides sufficient inhibition to start driving the winner-take-all. So whereas the inhibitory weights of -0.2 that worked for inputs ranging from 0.10 to 0.14 in the winner-take-all network in Figure 1a could not provide enough inhibition to drive a single winner when the inputs were over 0.5 (Figure 1b), the DAWTA regulatory unit simply increases its activation level until the inhibition it provides is sufficient to start suppressing the eventual losers (Figures 4a and 4b). As can also be seen in the figures, the activation of the regulatory unit tends to vary over time with different feedback from the network in a process that maximizes differentiation between units while assuring that the group of remaining potential winners stays active and are not over-inhibited.

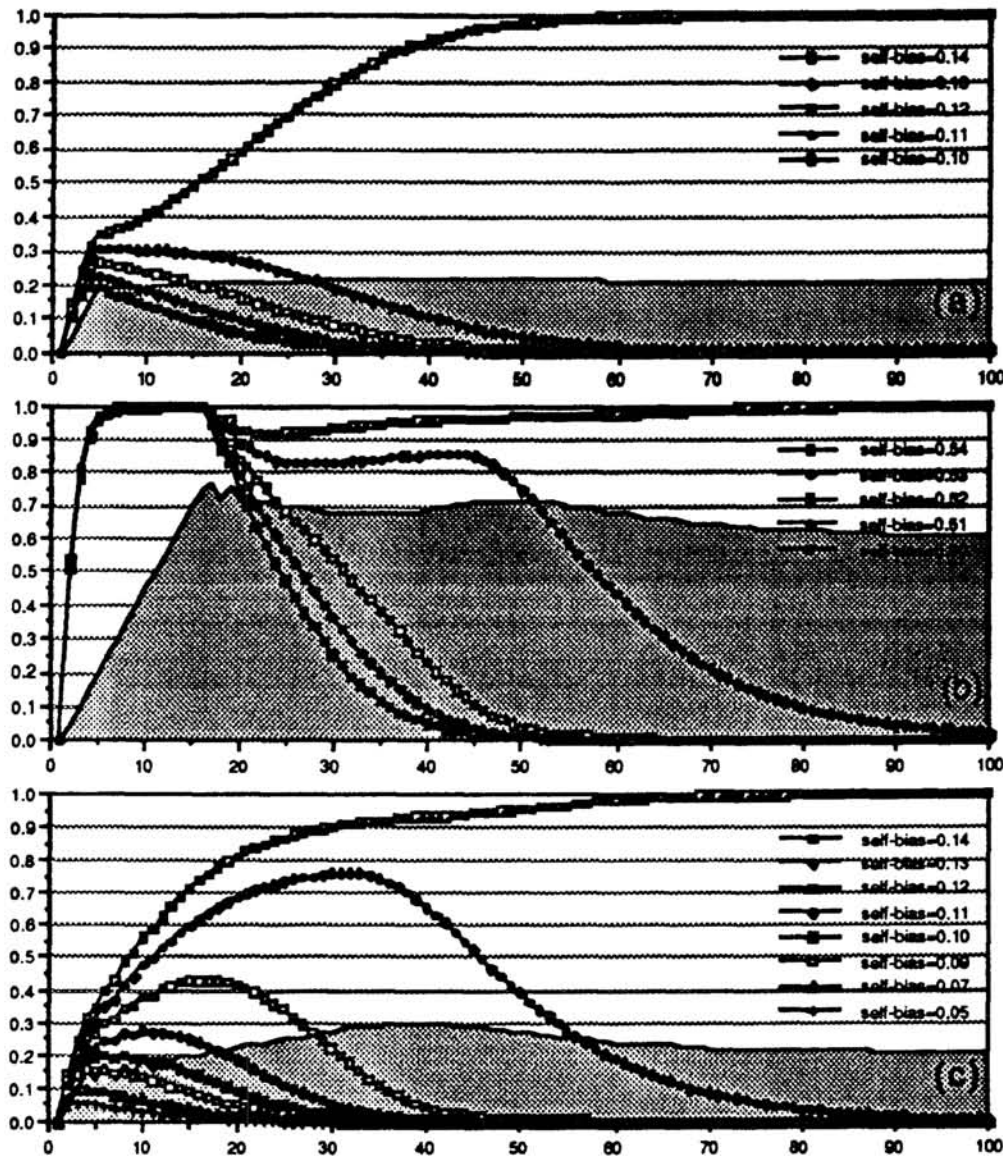

Figure 4. Plots of activation versus time in a dynamically-adaptive winner-take-all network given the same activation functions and initial conditions of the winner-take-all plots in Figure 1. The grey background plot shows the activation level of the regulatory unit. (a) With five units activated with self-biases from 0.10 to 0.14. (b) With five units activated with self-biases from 0.50 to 0.54. (c) With 100 units activated with self-biases from 0.01 to 0.14

Finally, though there is not space to show the graphic results here, the same DAWTA networks have been simulated to drive a successful winner-take-all within 200 cycles on networks ranging in size from 2 to 10,000 units and on initial conditions where the winning unit has an input of 0.000001 to initial conditions where the winning unit has an input of 0.999, without tuning the network in any way. The same networks have also been successfully simulated to act as $K$-winner-take-all networks (i.e. to select the $K$ most active units) by simply setting the desired value for $k$ in the DAWTA's self bias term $kw_d$.

## 4. CONCLUSIONS

We have presented Dynamically-Adaptive Winner-Take-All networks, which use $O(N)$ connections to perform the winner-take-all function. Unlike normal winner-take-all networks, DAWTA networks are able to select the most highly-activated unit out of a group of units for nearly any network size and initial condition without tuning any network parameters. They are able to do so because the inhibition that drives the winner-take-all network is provided by a regulatory-unit that is constantly getting feedback from the state of the network and dynamically adjusting its level to provide the right amount of inhibition to differentiate the winning unit from the losers. An important side-feature of this dynamically-adaptive inhibition approach is that it can be biased to select the $K$ most highly-activated units, and therefore serve as a $K$-winners-take-all network.

## Footnotes

[1]As in all winner-take-all networks, the competing units may also have inputs from outside the network that provide the initial activations driving the competition.

### References

Barnden, J. (1990). The power of some unusual connectionist data-structuring techniques. In J. A. Barnden and J. B. Pollack (Eds.), *Advances in connectionist and neural computation theory*, Norwood, NJ: Ablex.

Barnden, J., Kankanahalli, S., Dharmavaratha, D. (1990). Winner-take-all networks: Time-based versus activation-based mechanisms for various selection tasks. *Proceedings of the IEEE International Symposium on Circuits and Systems*, New Orleans, LA.

Feldman, J. A. & Ballard, D. H. (1982). Connectionist models and their properties. *Cognitive Science, 6*, 205-254.

Kohonen, T. (1984). *Self-organization and associative memory*. New York: Springer-Verlag, Berlin.

Lange, T. (1990). Simulation of heterogeneous neural networks on serial and parallel machines. *Parallel Computing, 14*, 287-303.

Lange, T. (in press). Hybrid connectionist models: Temporary bridges over the gap between the symbolic and the subsymbolic. To appear in J. Dinsmore (ed.), *Closing the Gap: Symbolic vs. Subsymbolic Processing*. Hillsdale, NJ: Lawrence Erlbaum Associates.

Lange, T. & Dyer, M. G. (1989a). Dynamic, non-local role-bindings and inferencing in a localist network for natural language understanding. In David S. Touretzky, editor, *Advances in Neural Information Processing Systems I*, p. 545-552, Morgan Kaufmann, San Mateo, CA.

Lange, T. & Dyer, M. G. (1989b). High-level inferencing in a connectionist network. *Connection Science, 1 (2)*, 181-217.

Majani, E., Erlanson, R. & Abu-Mostafa, Y. (1989). On the k-winners-take-all network. In David S. Touretzky, editor, *Advances in Neural Information Processing Systems I*, p. 634-642, Morgan Kaufmann, San Mateo, CA.

McClelland, J. L., & Rumelhart, D. E. (1981). An interactive activation model of context effects in letter perception: Part 1. An account of basic findings. *Psychological Review, 88*, 375-407.

Reggia, J. A. (1989). Methods for deriving competitive activation mechanisms. *Proceedings of the First Annual International Joint Conference on Neural Networks*.

Touretzky, D. (1989). Analyzing the energy landscapes of distributed winner-take-all networks (1989). In David S. Touretzky, editor, *Advances in Neural Information Processing Systems I*, p. 626-633, Morgan Kaufmann, San Mateo, CA.

Touretzky, D., & Hinton, G. (1988). A distributed connectionist production system. *Cognitive Science, 12*, 423-466.
